# On the infeasibility of training neural networks with small squared errors

**Van H. Vu**
Department of Mathematics, Yale University
vuha@math.yale.edu

## Abstract

We demonstrate that the problem of training neural networks with small (average) squared error is computationally intractable. Consider a data set of $M$ points $(X_i, Y_i)$, $i = 1, 2, \ldots, M$, where $X_i$ are input vectors from $R^d$, $Y_i$ are real outputs ($Y_i \in R$). For a network $f_0$ in some class $\mathcal{F}$ of neural networks, $(1/M) \sum_{i=1}^{M} (f_0(X_i) - Y_i)^2)^{1/2} - inf_{f \in \mathcal{F}} (1/M) \sum_{i=1}^{M} (f(X_i) - Y_i)^2)^{1/2}$ is the (avarage) relative error occurs when one tries to fit the data set by $f_0$. We will prove for several classes $\mathcal{F}$ of neural networks that achieving a relative error smaller than some fixed positive threshold (independent from the size of the data set) is NP-hard.

## 1  Introduction

Given a data set $(X_i, Y_i)$, $i = 1, 2, \ldots, M$, $X_i$ are input vectors from $R^d$, $Y_i$ are real outputs ($Y_i \in R$). We call the points $(X_i, Y_i)$ *data points*. The training problem for neural networks is to find a network from some class (usually with fixed number of nodes and layers), which fits the data set with small error. In the following we describe the problem with more details.

Let $\mathcal{F}$ be a class (set) of neural networks, and $\alpha$ be a metric norm in $R^M$. To each $f \in \mathcal{F}$, associate an *error vector* $E_f = (|f(X_i) - Y_i|)_{i=1}^{M}$ ($E_{\mathcal{F}}$ depends on the data set, of course, though we prefer this notation to avoid difficulty of having too many subindices). The norm of $E_f$ in $\alpha$ shows how well the network $f$ fits the data regarding to this particular norm. Furthermore, let $e_{\alpha, \mathcal{F}}$ denote the smallest error achieved by a network in $\mathcal{F}$, namely:

$$e_{\alpha, \mathcal{F}} = \min_{f \in \mathcal{F}} \|E_f\|_\alpha$$

In this context, the training problem we consider here is to find $f \in \mathcal{F}$ such that

$\|E_f\|_\alpha - e_{\alpha,\mathcal{F}} \leq \epsilon_{\mathcal{F}}$, where $\epsilon_{\mathcal{F}}$ is a positive number given in advance, and does not depend on the size $M$ of the data set. We will call $\epsilon_{\mathcal{F}}$ *relative error*. The norm $\alpha$ is chosen by the nature of the training process, the most common norms are:

$l_\infty$ norm: $\|v\|_\infty = max|v_i|$ (interpolation problem)

$l_2$ norm: $\|v\|_2 = (1/M \sum_{i=1}^M v_i^2)^{1/2}$, where $v = (v_i)_{i=1}^M$ (least square error problem).

The quantity $\|E_f\|_{l_2}$ is usually referred to as the *emperical error* of the training process. The first goal of this paper is to show that achieving small emperical error is NP-hard. From now on, we work with $l_2$ norm, if not otherwise specified.

A question of great importance is: given the data set, $\mathcal{F}$ and $\epsilon_{\mathcal{F}}$ in advance, could one find an efficient algorithm to solve the training problem formulated above. By efficiency we mean an algorithm terminating in polynomial time (polynomial in the size of the input). This question is closely related to the problem of learning neural networks in polynomial time (see [3]). The input in the algorithm is the data set, by its size we means the number of bits required to write down all $(X_i, Y_i)$.

**Question 1.** *Given $\mathcal{F}$ and $\epsilon_{\mathcal{F}}$ and a data set. Could one find an efficient algorithm which produces a function $f \in \mathcal{F}$ such that $\|E_f\| < e_{\mathcal{F}} + \epsilon_{\mathcal{F}}$*

Question 1 is very difficult to answer in general. In this paper we will investigate the following important sub-question:

**Question 2.** *Can one achieve arbitrary small relative error using polynomial algorithms ?*

Our purpose is to give a negative answer for Question 2. This question was posed by L. Jones in his seminar at Yale (1996). The crucial point here is that we are dealing with $l_2$ norm, which is very important from statistical point of view. Our investigation is also inspired by former works done in [2], [6], [7], etc, which show negative results in the $l_\infty$ norm case.

**Definition.** *A positive number $\epsilon$ is a threshold of a class $\mathcal{F}$ of neural networks if the training problem by networks from $\mathcal{F}$ with relative error less than $\epsilon$ is NP-hard (i.e., computationally infeasible).*

In order to provide a negative answer to Question 2, we are going to show the existence of thresholds (which is independent from the size of the data set) for the following classes of networks.

- $\mathcal{F}_n = \{f | f(x) = (1/n)(\sum_{i=1}^n \text{step}(a_i x - b_i))\}$
- $\mathcal{F}'_n = \{f | f(x) = (\sum_{i=1}^n c_i \text{step}(a_i x - b_i))\}$
- $\mathcal{G}_n = \{g | g(x) = \sum_{i_1}^n c_i \phi_i(a_i x - b_i)\}$

where $n$ is a positive integer, $\text{step}(x) = 1$ if $x$ is positive and zero otherwise, $a_i$ and $x$ are vectors from $R^d$, $b_i$ are real numbers, and $c_i$ are positive numbers. It is clear that the class $\mathcal{F}'_n$ contains $\mathcal{F}_n$; the reason why we distinguish these two cases is that the proof for $\mathcal{F}_n$ is relatively easy to present, while contains the most important ideas. In the third class, the functions $\phi_i$ are sigmoid functions which satisfy certain Lipchitzian conditions (for more details see [9])

**Main Theorem**

*(i) The classes $\mathcal{F}_1$, $\mathcal{F}_2$, $\mathcal{F}'_2$ and $\mathcal{G}_2$ have absolute constant (positive) thresholds*

*(ii) For every class $\mathcal{F}_{n+2}, n > 0$, there is a threshold of form $\zeta n^{-3/2} d^{-1/2}$.*

*(iii) For every $\mathcal{F}'_{n+2}, n > 0$, there is a threshold of form $\zeta n^{-3/2} d^{-3/2}$.*

*(iv) For every class $\mathcal{G}_{n+2}, n > 0$, there is a threshold of form $\zeta n^{-5/2} d^{-1/2}$.*

*In the last three statements, $\zeta$ is an absolute positive constant.*

Here is the key argument of the proof. Assume that there is an algorithm A which solves the training problem in some class (say $\mathcal{F}_n$ ) with relative error $\epsilon$. From some (properly chosen) NP-hard problem. we will construct a data set so that if $\epsilon$ is sufficiently small, then the solution found by A (given the constructed data set as input) in $\mathcal{F}_n$ implies a solution for the original NP-hard problem. This will give a lower bound on $\epsilon$, if we assume that the algorithm A is polynomial. In all proofs the leading parameter is $d$ (the dimension of data inputs). So by polynomial we mean a polynomial with $d$ as variable. All the input (data) sets constructed will have polynomial size in $d$.

The paper is organized as follow. In the next Section, we discuss earlier results concerning the $l_\infty$ norm. In Section 3, we display the NP-hard results we will use in the reduction. In Section 4, we prove the main Theorem for class $\mathcal{F}_2$ and mention the method to handle more general cases. We conclude with some remarks and open questions in Section 5.

To end this Section, let us mention one important corollary. The Main Theorem implies that learning $\mathcal{F}_n$, $\mathcal{F}'_n$ and $\mathcal{G}_n$ (with respect to $l_2$ norm) is hard. For more about the connection between the complexity of training and learning problems, we refer to [3], [5].

**Notation:** Through the paper $U_d$ denotes the unit hypercube in $R^d$. For any number $x$, $x_d$ denotes the vector $(x, x, ... x)$ of length $d$. In particular, $0_d$ denotes the origin of $R^d$. For any half space $H$, $\bar{H}$ is the complement of $H$. For any set $A$, $|A|$ is the number of elements in $A$. A function $y(d)$ is said to have order of magnitude $\Theta(F(d))$, if there are $c < C$ positive constants such that $c < y(d)/F(d) < C$ for all $d$.

## 2   Previous works in the $l_\infty$ case

The case $\alpha = l_\infty$ (interpolation problem) was considered by several authors for many different classes of (usually) 2-layer networks (see [6],[2], [7], [8]). Most of the authors investigate the case when there is a perfect fit, i.e., $\epsilon_{l_\infty}.\mathcal{F} = 0$. In [2], the authors proved that training 2-layer networks containing 3 step function nodes with zero relative error is NP-hard. Their proof can be extended for networks with more inner nodes and various logistic output nodes. This generalized a former result of Maggido [8] on data set with rational inputs. Combining the techniques used in [2] with analysis arguments, Lee Jones [6] showed that the training problem with relative error 1/10 by networks with two monotone Lipschitzian Sigmoid inner nodes and linear output node, is also NP-hard (NP-complete under certain circumstances). This implies a threshold (in the sense of our definition) $(1/10)M^{-1/2}$ for the class examined. However, this threshold is rather weak, since it is decreasing in $M$. This result was also extended for the $n$ inner nodes case [6].

It is also interesting to compare our results with Judd's. In [7] he considered the following problem "Given a network and a set of training examples (a data set), does there exist a set of weights so that the network gives correct output for all training examples ?" He proved that this problem is NP-hard even if the network is

required to produce the correct output for two-third of the traing examples. In fact, it was shown that there is a class of networks and a data sets so that any algorithm will produce poorly on some networks and data sets in the class. However, from this result one could not tell if there is a network which is "hard to train" for all algorithms. Moreover, the number of nodes in the networks grows with the size of the data set. Therefore, in some sense, the result is not independent from the size of the data set.

In our proofs we will exploit many techniques provided in these former works. The crucial one is the reduction used by A. Blum and R. Rivest, which involves the NP-hardness of the Hypergraph 2-Coloring problem.

## 3   Some NP hard problems

**Definition** *Let $B$ be a CNF formula, where each clause has at most $k$ literals. Let $max(B)$ be the maximum number of clauses which can be satisfied by a truth assignment. The APP MAX k-SAT problem is to find a truth assignment which satisfies $(1 - \epsilon)max(B)$ clauses.*

The following Theorem says that this approximation problem is NP -hard, for some small $\epsilon$.

**Theorem 3.1.1** *Fix $k \geq 2$. There is $\epsilon_1 > 0$, such that finding a truth assignment. which satisfies at least $(1 - \epsilon_1)max(B)$ clauses is NP-hard.*

The problem is still hard, when every literal in $B$ appears in only few clauses, and every clause contains only few literals. Let $\mathcal{B}_3(5)$ denote the class of CNFs with at most 3 literals in a clause and every literal appears in at most 5 clauses (see [1]).

**Theorem 3.1.2** *There is $\epsilon_2 > 0$ such that finding a truth assignment, which satisfies at least $(1 - \epsilon)max(B)$ clauses in a formula $B \in \mathcal{B}_3(5)$ is NP-hard.*

The optimal thresholds in these theorems can be computed, due to recent results in Thereotical Computer Science. Because of space limitation, we do not go into this matter.

Let $H = (V, E)$ be a hypergraph on the set $V$, and $E$ is the set of edges (collection of subsets of $V$). Elements of $V$ are called vertices. The degree of a vertex is the number of edges containing the vertex. We could assume that each edge contains at least two vertices. Color the vertices with color Blue or Red. An edge is *colorful* if it contains vertices of both colors, otherwise we call it *monochromatic*. Let $c(H)$ be the maximum number of colorful edges one can achieve by a coloring. By a probabilistic argument, it is easy to show that $c(H)$ is at least $|E|/2$ (in a random coloring, an edge will be colorful with probability at least $1/2$). Using 3.1.2, we could prove the following theorem (for the proof see [9])

**Theorem 3.1.3** *There is a constant $\epsilon_3 > 0$ such that finding a coloring with at least $(1 - \epsilon_3)c(H)$ colorful edges is NP-hard. This statement holds even in the case when every but one degree in $H$ is at most 10*

## 4   Proof for $\mathcal{F}_2$

We follow the reduction used in [2]. Consider a hypergraph $H(V, E)$ described Theorem 3.2.1. Let $V = \{1, 2, \ldots, d + 1\}$, where with the possible exception of the vertex $d + 1$, all other vertices have degree at most 10. Every edge will have at least 2 and at most 4 vertices. So the number of edges is at least $(d + 1)/4$.

Let $p_i$ be the $i^{th}$ unit vector in $R^{d+1}$, $p_i = (0, 0, \ldots, 0, 1, 0, \ldots, 0)$. Furthermore, $\chi_C = \sum_{i \in C} p_i$ for every edge $C \in E$. Let $S$ be a coloring with maximum number of colorful edges. In this coloring denote by $A_1$ the set of colorful edges and by $A_2$ the set of monochromatic edges. Clearly $|A_1| = c(H)$.

Our data set will be the following (inputs are from $R^{d+1}$ instead of from $R^d$, but it makes no difference)

$$D = \{(p_i, 1/2)\}_{i=1}^d \cup \{(p_{d+1}, 1/2)^t\} \cup \{(0_{d+1}, 1)^t\} \cup \{(\chi_C, 1)|C \in A_1\} \cup \{(\chi_C, 1/2)|C \in A_2\}$$

where $(p_{d+1}, 1/2)^t$ and $(0_{d+1}, 1)^t$ means $(p_{d+1}, 1/2)$ and $(0_{d+1}, 1)$ are repeated $t$ times in the data set, resp. Similarly to [2], consider two vectors $a$ and $b$ in $R^{d+1}$ where

$$a = (a_1, \ldots, a_{d+1}), a_i = -1 \text{ if } i \text{ is Red and } a_i = d + 1 \text{ otherwise}$$
$$b = (b_1, \ldots, b_{d+1}), b_i = -1 \text{ if } i \text{ is Blue and } b_i = d + 1 \text{ otherwise}$$

It is not difficult to verify that the function $f_0 = (1/2)(\text{step}\,(ax + 1/2) + \text{step}\,(bx + 1/2))$ fits the data perfectly, thus $e_{\mathcal{F}_2} = \|E_{f_0}\| = 0$.

Suppose $f = (1/2)(\text{step}\,(cx - \gamma) + \text{step}\,(dx - \delta))$ satisfies

$$M\|E_f\|^2 = \sum_{i=1}^M (f(X_i) - Y_i)^2 < M\epsilon^2$$

Since if $f(X_i) \neq Y_i$ then $(f(X_i) - Y_i)^2 \geq 1/4$, the previous inequality implies: $\mu_0 = |\{i, f(X_i) \neq Y_i\}| < 4M\epsilon^2 = \mu$

The ratio $\mu_0/M$ is called misclassification ratio, and we will show that this ratio cannot be arbitrary small. In order to avoid unnecessary ceiling and floor symbols, we assume the upper-bound $\mu$ is an integer. We choose $t = \mu$ so that we can also assume that $(0_{d+1}, 1)$ and $(p_{d+1}, 1/2)$ are well classified. Let $H_1$ ($H_2$) be the half space consisting of $x$: $cx - \gamma > 0$ ($dx - \delta > 0$). Note that $0_d \in H_1 \cap H_2$ and $p_{d+1} \in \bar{H}_1 \cup \bar{H}_2$. Now let $P_1$ denote the set of $i$ where $p_i \notin H_1$, and $P_2$ the set of $i$ such that $p_i \in H_1 \cap H_2$. Clearly, if $j \in P_2$, then $f(p_j) \neq Y_j$, hence: $|P_2| \leq \mu$. Let $Q = \{C \in E | C \cap P_2 \neq \emptyset\}$. Note that for each $j \in P_2$, the degree of $j$ is at most 10, thus: $|Q| \leq 10|P_2| \leq 10\mu$

Let $A_1' = \{C | f(\chi_C) = 1\}$. Since less than $\mu$ points are misclassified, $|A_1' \triangle A_1| < \mu$. Color $V$ by the following rule: (1) if $p_i \in P_1$, then $i$ is Red; (2) if $p_i \in P_2$, color $i$ arbitrarily, either Red or Blue; (3) if $p_i \notin P_1 \cup P_2$, then $i$ is Blue.

Now we can finish the proof by the following two claims:

**Claim 1:** *Every edge in $A_1' \backslash Q$ is colorful.* It is left to readers to verify this simple statement.

**Claim 2:** $|A_1' \backslash Q|$ *is close to* $|A_1|$.

Notice that:

$$|A_1 \backslash (A_1' \backslash Q)| \leq |A_1 \triangle A_1'| + |Q| \leq \mu + 10\mu = 11\mu$$

Observe that the size of the data set is $M = d + 2t + |E|$, so $|E| + d \geq M - 2t = M - 2\mu$. Moreover, $|E| \geq (d + 1)/4$, so $|E| \geq (1/5)(M - 2\mu)$. On the other hand, $|A_1| \geq (1/2)|E|$, all together we obtain; $|A_1| \geq (1/10)(M - \mu)$, which yields:

$$|A_1'\backslash Q| \geq |A_1|(1 - 11\frac{\mu}{|A_1|}) \geq |A_1|(1 - 110(\frac{\mu}{(M-\mu)}))$$

$$\geq |A_1|(1 - 110(\frac{4\epsilon^2}{1-4\epsilon^2})) = |A_1|(1 - k(\epsilon))$$

Choose $\epsilon = \epsilon_4$ such that $k(\epsilon_4) \leq \epsilon_3$ (see Theorem 3.1.3). Then $\epsilon_4$ will be a threshold for the class $\mathcal{F}_2$. This completes the proof. Q.E.D.

Due to space limitation, we omit the proofs for other classes and refer to [9]. However, let us at least describe (roughly) the general method to handle these cases. The method consists of following steps:

• Extend the data set in the previous proof by a set of (special) points.

• Set the multiplicities of the special points sufficiently high so that those points should be well-classified.

• If we choose the special points properly, the fact that these points are well-classified will determine (roughly) the behavior of all but 2 nodes. In general we will show that all but 2 nodes have little influence on the outputs of non-special data points.

• The problem basically reduces to the case of two nodes. By modifying the previous proof, we could achieve the desired thresholds.

## 5    Remarks and open problems

• Readers may argue about the existence of (somewhat less natural) data points of high multiplicities. We can avoid using these data points by a combinatorial trick described in [9].

• The proof in Section 4 could be carried out using Theorem 3.1.2. However, we prefer using the hypergraph coloring terminology (Theorem 3.1.3), which is more convenient and standard. Moreover, Theorem 3.1.3 itself is interesting, and has not been listed among well known "approximation is hard" theorems.

• It remains an open question to determine the right order of magnitude of thresholds for all the classes we considered. (see Section 1). By technical reasons, in the Main theorem, the thresholds for more than two nodes involve the dimension ($d$). We conjecture that there are dimension-free thresholds.

**Acknowledgement** We wish to thank A. Blum, A. Barron and L. Lovász for many useful ideas and discussions.

## References

[1] S. Arora and C. Lund *Hardness of approximation*, book chapter, preprint

[2] A. Blum, R. Rivest *Training a 3-node neural network is NP-hard* Neutral Networks, Vol 5., p 117-127, 1992

[3] A. Blumer, A. Ehrenfeucht, D. Haussler, M. Warmuth, *Learnability and the Vepnik-Chervonenkis Dimension*, Journal of the Association for computing Machinery, Vol 36, No. 4, 929-965, 1989.

[4] M. Garey and D. Johnson, Computers and intractability: A guide to the theory of NP-completeness, San Francisco, W.H.Freeman, 1979

[5] D. Haussler, *Generalizing the PAC model for neural net and other learning applications* (Tech. Rep. UCSC-CRL-89-30). Santa Cruz. CA: University of California 1989.

[6] L. Jones, *The computational intractability of training sigmoidal neural networks* (preprint)

[7] J. Judd *Neutral Networks and Complexity of learning*, MIT Press 1990.

[8] N. Meggido, *On the complexity of polyhedral separability* (Tech. Rep. RJ 5252) IBM Almaden Research Center, San Jose, CA

[9] V. H. Vu, *On the infeasibility of training neural networks with small squared error*, manuscript.